# Learning How To Teach
## or
## Selecting Minimal Surface Data

**Davi Geiger**
Siemens Corporate Research, Inc
755 College Rd. East
Princeton, NJ 08540
USA

**Ricardo A. Marques Pereira**
Dipartimento di Informatica
Universita di Trento
Via Inama 7, Trento, TN 38100
ITALY

## Abstract

Learning a map from an input set to an output set is similar to the problem of reconstructing hypersurfaces from sparse data (Poggio and Girosi, 1990). In this framework, we discuss the problem of automatically selecting "minimal" surface data. The objective is to be able to approximately reconstruct the surface from the selected sparse data. We show that this problem is equivalent to the one of compressing information by data removal and the one of learning how to teach. Our key step is to introduce a process that statistically selects the data according to the model. During the process of data selection (learning how to teach) our system (teacher) is capable of predicting the new surface, the approximated one provided by the selected data. We concentrate on piecewise smooth surfaces, e.g. images, and use mean field techniques to obtain a deterministic network that is shown to compress image data.

## 1 Learning and surface reconstruction

Given a dense input data that represents a hypersurface, how could we automatically select *very* few data points such as to be able to use these fewer data points (sparse data) to approximately reconstruct the hypersurface ?

We will be using the term surface to refer to hypersurface (surface in multidimen-

sions) throughout the paper.

It has been shown (Poggio and Girosi, 1990) that the problem of reconstructing a surface from sparse and noisy data is equivalent to the problem of learning from examples. For instance, to learn how to add numbers can be cast as finding the map from $X = \{pair\, of\, numbers\}$ to $F = \{sum\}$ from a set of noisy examples. The surface is $F(X)$ and the sparse and noisy data are the set of $N$ examples $\{(x_i, d_i)\}$, where $i = 0, 1, ..., N$ and $x_i = (a_i, b_i) \in X$, such that $a_i + b_i = d_i + \eta_i$ ($\eta_i$ being the noise term). Some *a priori* information about the surface, e.g. the smoothness one, is necessary for reconstruction.

Consider a set of $N$ input-output examples, $\{(x_i, d_i)\}$, and a form $\| Pf \|^2$ for the cost of the deviation of $f$, the approximated surface, from smoothness. $P$ is a differential operator and $\| \cdot \|$ is a norm (usually $L^2$). To find the surface $f$, that best fits (i) the data and (ii) the smoothness criteria, is to solve the problem of minimizing the functional

$$V(f) = \sum_{i=0}^{N-1} (d_i - f(x_i))^2 + \mu \| Pf \|^2 \ .$$

Different methods of solving the function can yield different types of network. In particular using the Green's method gives supervised backprop type of networks (Poggio and Girosi, 1990) and using optimization techniques (like gradient descent) we obtain unsupervised (with feedback) type of networks.

## 2  Learning how to teach arithmetic operations

The problem of learning how to *add* and *multiply* is a simple one and yet provide insights to our approach of selecting the minimum set of examples.

**Learning arithmetic operations**   The surface given by the addition of two numbers, namely $f(x, y) = x + y$, is a plane passing through the origin. The multiplication surface, $f(x, y) = xy$, is hyperbolic. The *a priori* knowledge of the *addition* and *multiplication* surface can be expressed as a minimum of the functional

$$V(f) = \int_{-\infty}^{\infty} \int_{-\infty}^{\infty} \| \nabla^2 f(x, y) \| \, dx \, dy$$

where

$$\nabla^2 f(x, y) = (\frac{\partial^2}{\partial x^2} + \frac{\partial^2}{\partial y^2}) f(x, y) \quad .$$

Other functions also minimize $V(f)$, like $f(x, y) = x^2 - y^2$, and so a few examples are necessary to learn how to add and multiply given the above prior knowledge. If the prior assumption consider a larger class of basis functions, then more examples will be required. Given $p$ input-output examples, $\{(x_i, y_i); d_i\}$, the learning problem of adding and multiplying can be cast as the optimization of

$$V(f) = \sum_{i=0}^{p-1}(f(x_i, y_i) - d_i)^2 + \mu \int_{-\infty}^{\infty}\int_{-\infty}^{\infty} \| \nabla^2 f(x,y) \| \, dx \, dy \quad .$$

We now consider the problem of selecting the examples from the full surface data.

**A sparse process for selecting data**    Let us assume that the full set of data is given in a 2-Dimensional lattice. So we have a finite amount of data ($N^2$ data points), with the input-output set being $\{(x_i, y_j); d_{ij}\}$, where $i, j = 0, 1, ..., N-1$. To select $p$ examples we introduce a sparse process that selects out data by modifying the cost function according to

$$V = \sum_{i,j=0}^{N-1}(1-s_{ij})(f(x_i,y_j)-d_{ij})^2 + \mu \int_{-\infty}^{\infty}\int_{-\infty}^{\infty} \| \nabla^2 f(x,y) \| + \lambda(p - \sum_{i,j=0}^{N-1}(1-s_{ij}))^2$$

where $s_{ij} = 1$ selects out the data and we have added the last term to assure that $p$ examples are selected. The data term forces noisy data to be thrown out first, the second order smoothness of $f$ reduces the need for many examples ($p \approx 10$) to learn these arithmetic operations. Learning $s$ is equivalent to learn how to select the examples, or to learn how to teach. The system (teacher) has to learn a set of examples (sparse data) that contains all the "relevant" information. The redundant information can be "filled in" by the prior knowledge. Once the teacher has learned these selected examples, he, she or it (machine) presents them to the student that with the *a priori* knowledge about surfaces is able to approximately learn the full input-output map (surface).

# 3    Teaching piecewise smooth surfaces

We first briefly introduce the weak membrane model, a coupled Markov random field for modeling piecewise smooth surfaces. Then we lay down the framework for learning to teach this surface.

## 3.1    Weak membrane model

Within the Bayes approach the *a priori* knowledge that surfaces are smooth (first order smoothness) but not at the discontinuities has been analyzed by (Geman and Geman, 1984) (Blake and Zisserman, 1987) (Mumford and Shah, 1985) (Geiger and Girosi, 1991). If we consider the noise to be white Gaussian, the final posterior probability becomes $P(f, l|g) = \frac{1}{Z}e^{-\beta V(f,l)}$, where

$$V(f,l) = \sum_{i,j}[(f_{ij} - g_{ij})^2 + \mu \| \nabla f \|_{ij}^2 (1 - l_{ij}) + \gamma_{ij} l_{ij}], \tag{1}$$

We represented surfaces by $f_{ij}$ at pixel $(i,j)$, and discontinuities by $l_{ij}$. The input data is $g_{ij}$, $\| \nabla f \|_{ij}$ is the norm of the gradient at pixel $(i,j)$. $Z$ is a normalization

constant, known as the partition function. $\beta$ is a global parameter of the model and is inspired on thermodynamics, and $\mu$ and $\gamma_{ij}$ are parameters to be estimated. This model, when used for image segmentation, has been shown to give a good pattern of discontinuities and eliminate the noise. Thus, suggesting that the piecewise assumption is valid for images.

## 3.2   Redundant data

We have assumed the surface to be smooth and therefore there is redundant information within smooth regions. We then propose a model that selects the "relevant" information according to two criteria

1.  **Discontinuity data:**   Discontinuities usually capture relevant information, and it is possible to roughly approximate surfaces just using edge data (see Geiger and Pereira, 1990). A limitation of just using edge data is that an oversmoothed surface is represented.

2. **Texture data:**   Data points that have significant gradients (not enough to be a discontinuity) are here considered texture data. Keeping texture data allows us to distinguish between flat surfaces, as for example a clean sky in an image, and texture surfaces, as for example the leaves in the tree (see figure 2).

## 3.3   The sparse process

Again, our proposal is first to extend the weak membrane model by including an additional binary field - the *sparse process s*- that is 1 when data is selected out and 0 otherwise. There are natural connections between the process $s$ and robust statistics (Huber, 1988) as discussed in (Geiger and Yuille, 1990) and (Geiger and Pereira, 1991). We modify (1) by considering (see also Geiger and Pereira, 1990)

$$V(f, l, s) = \sum_{i,j} [(1 - s_{ij})(f_{ij} - g_{ij})^2 + \mu \parallel \nabla f \parallel_{ij}^2 (1 - l_{ij}) + \eta_{ij} s_{ij} + \gamma_{ij} l_{ij}]. \quad (2)$$

where we have introduced the term $\eta_{ij} s_{ij}$ to keep some data otherwise $s_{ij} = 1$ everywhere. If the data term is too large, the process $s = 1$ can suppress it. We will now assume that the data is noise-free, or that the noise has already been smoothed out. We then want to find which data points ($s = 0$) are necessary to keep to reconstruct $f$.

## 3.4   Mean field equations and unsupervised networks

To impose the **discontinuity data** constraint we use the hard constraint technique (Geiger and Yuille, 1990 and its references). We do *not allow* states that throw out data ($s_{ij} = 1$) at the edge location ($l_{ij} = 1$). More precisely, within the statistical framework we reduce the possible states for the processes $s$ and $l$ to $s_{ij} l_{ij} = 0$. Therefore, excluding the state ($s_{ij} = 1, l_{ij} = 1$). Applying the saddle point approximation, a well known mean field technique (Geiger and Girosi, 1989 and its references), on the field $f$, we can compute the partition function

$$Z = \sum_{f=(0,..,255)^{N^2}} \sum_{s,l=(0,1)^{N^2}}^{s.l=0} e^{-\beta V(f,l,s)} \approx \sum_{s,l=(0,1)^{N^2}}^{s.l=0} e^{-\beta V(\bar{f},l,s)} \approx \prod_{ij} Z_{ij}$$

$$Z_{ij} = \left(e^{-\beta[\gamma_{ij}+(\bar{f}_{ij}-g_{ij})^2]} + e^{-\beta[\mu\|\nabla f\|^2_{ij}+\eta_{ij}]} + e^{-\beta[\mu\|\nabla f\|^2_{ij}+(f_{ij}-g_{ij})^2]}\right) \qquad (3)$$

where $\bar{f}$ maximizes $Z$. After applying mean field techniques we obtain the following equations for the processes $l$ and $s$

$$\bar{l}_{ij} = e^{-\beta[\gamma_{ij}+(f_{ij}-g_{ij})^2]}/Z_{ij} \quad , \quad \bar{s}_{ij} = e^{-\beta[\mu\|\nabla f\|^2_{ij}+\eta_{ij}]}/Z_{ij} \qquad (4)$$

and, using the definition $\| \nabla f \|^2_{ij} = [(f_{i,j+1}-f_{i+1,j})^2 + (f_{i+1,j+1}-f_{i,j})^2$ , the mean field self consistent equation (Geiger and Pereira, 1991) becomes

$$(1-\bar{s}_{ij})(\bar{f}_{ij}-g_{ij}) = -\mu\Big\{K_{ij}(1-\bar{l}_{ij}) + K_{i-1,j-1}(1-\bar{l}_{i-1,j-1}) +$$

$$M_{i-1,j}(1-\bar{l}_{i-1,j}) + M_{i,j-1}(1-\bar{l}_{i,j-1})\Big\} \qquad (5)$$

where $K_{ij} = (f_{i+1,j+1} - f_{i,j})^2$ and $M_{ij} = (f_{i+1,j} - f_{i,j+1})^2$. The set of coupled equations (5) (4) can be mapped to an unsupervised network, we call a minimal surface representation network (MSRN), and can efficiently be solved in a massively parallel machine. Notice that $s_{ij} + l_{ij} \geq 1$, because of the hard constraint, and in the limit of $\beta \to \infty$ the processes $s$ and $l$ becomes either 0 or 1. In order to throw away redundant (smooth) data keeping some of the texture we adapt the cost $\eta_{ij}$ according to the gradient of the surface. More precisely, we set

$$\eta_{ij} = \eta[(\Delta^h_{ij}g)^2 + (\Delta^v_{ij}g)^2] \qquad (6)$$

where $(\Delta^h_{ij}g)^2 = (g_{i+1,j} - g_{i-1,j})^2$ and $(\Delta^v_{ij}g)^2 = (g_{i,j+1} - g_{i,j-1})^2$. The smoother is the data the lower is the cost to discard the data ($s_{ij} = 1$). In the limit of $\eta \to 0$ only edge data ($l_{ij} = 1$) is kept, since from (4) $lim_{\eta \to 0}s_{ij} = 1 - l_{ij}$.

### 3.5   Learning how to teach and the approximated surface

With the mean field equations we compute the approximated surface $f$ simultaneously to $s$ and to $l$. Thus, while learning the process $s$ (the selected data) the system also predict the approximated surface $f$ that the student will learn from the selected examples. By changing the parameters, say $\mu$ and $\eta$, the teacher can choose the optimal parameters such as to select less data and preserve the quality of the approximated surface. Once $s$ has been learned the system only feeds the selected data points to the learner machinery. We actually relax the condition and feed the learner with the selected data and the corresponding discontinuity map ($l$). Notice that in the limit of $\eta \to 0$ the selected data points are coincident with the discontinuities ($l = 1$).

# 4 Results: Image compression

We show the results of the algorithm to learn the minimal representation of images. The algorithm is capable of image compression and one advantage over the cosine transform (traditional method) is that it does not have the problem of breaking the images into blocks. However, a more careful comparison is needed.

## 4.1 Learning $s$, $f$, and $l$

To analyze the quality of the surface approximation, we show in figure 1 the performance of the network as we vary the threshold $\eta$. We first show a face image and the line process and then the predicted approximated surfaces together with the correspondent sparse process $s$.

## 4.2 Reconstruction, Generalization or "The student performance"

We can now test how the student learns from the selected examples, or how good is the surface reconstruction from the selected data. We reconstruct the approximate surfaces by running (5) again, but with the selected surface data points ($s_{ij} = 0$) and the discontinuities ($l_{ij} = 1$) given from the previous step. We show in figure 2f that indeed we obtain the predicted surfaces (the student has learned).

**References :**

E. B. Baum and Y. Lyuu. 1991. The transition to perfect generalization in perceptrons, *Neural Computation*, vol.3, no.3. pp.386-401.

A. Blake and A. Zisserman. 1987. Visual Reconstruction, *MIT Press*, Cambridge, Mass.

D. Geiger and F. Girosi. 1989. Coupled Markov random fields and mean field theory, Advances in Neural Information Processing Systems 2, Morgan Kaufmann, D. Touretzky.

D. Geiger and A. Yuille. 1991. A common framework for image segmentation, *Int. Jour. Comp. Vis.*,vol.6:3, pp. 227-243.

D. Geiger and F. Girosi. 1991. Parallel and deterministic algorithms for MRFs: surface reconstruction, *PAMI*, May 1991, vol.PAMI-13, 5, pp.401-412 .

D. Geiger and R. M. Pereira. 1991. The outlier process, *IEEE Workshop on Neural Networks for signal Processing*, Princeton, NJ.

S. Geman and D. Geman. 1984. Stochastic Relaxation, Gibbs Distributions, and the Bayesian Restoration of Images,*PAMI*, vol.PAMI-6, pp.721–741K.

J.J. Hopfield. 1984. Neural networks and physical systems with emergent collective computational abilities, *Proc. Nat. Acad. Sci.*,79 , pp. 2554-2558.

P.J. Huber. 1981. Robust Statistics, *John Wiley and Sons*, New York.

D. Mumford and J. Shah. 1985. Boundary detection by minimizing functionals, I , *Proc. IEEE Conf. on Computer Vision & Pattern Recognition*, San Francisco, CA .

T. Poggio and F. Girosi. 1990. Regularization algorithms for learning that are equivalent to multilayer network, *Science*,vol-247, pp. 978-982.

D. E. Rumelhart, G. Hinton and R. J. Willians. 1986. Learning internal representations by error backpropagation. *Nature*, 323, 533.

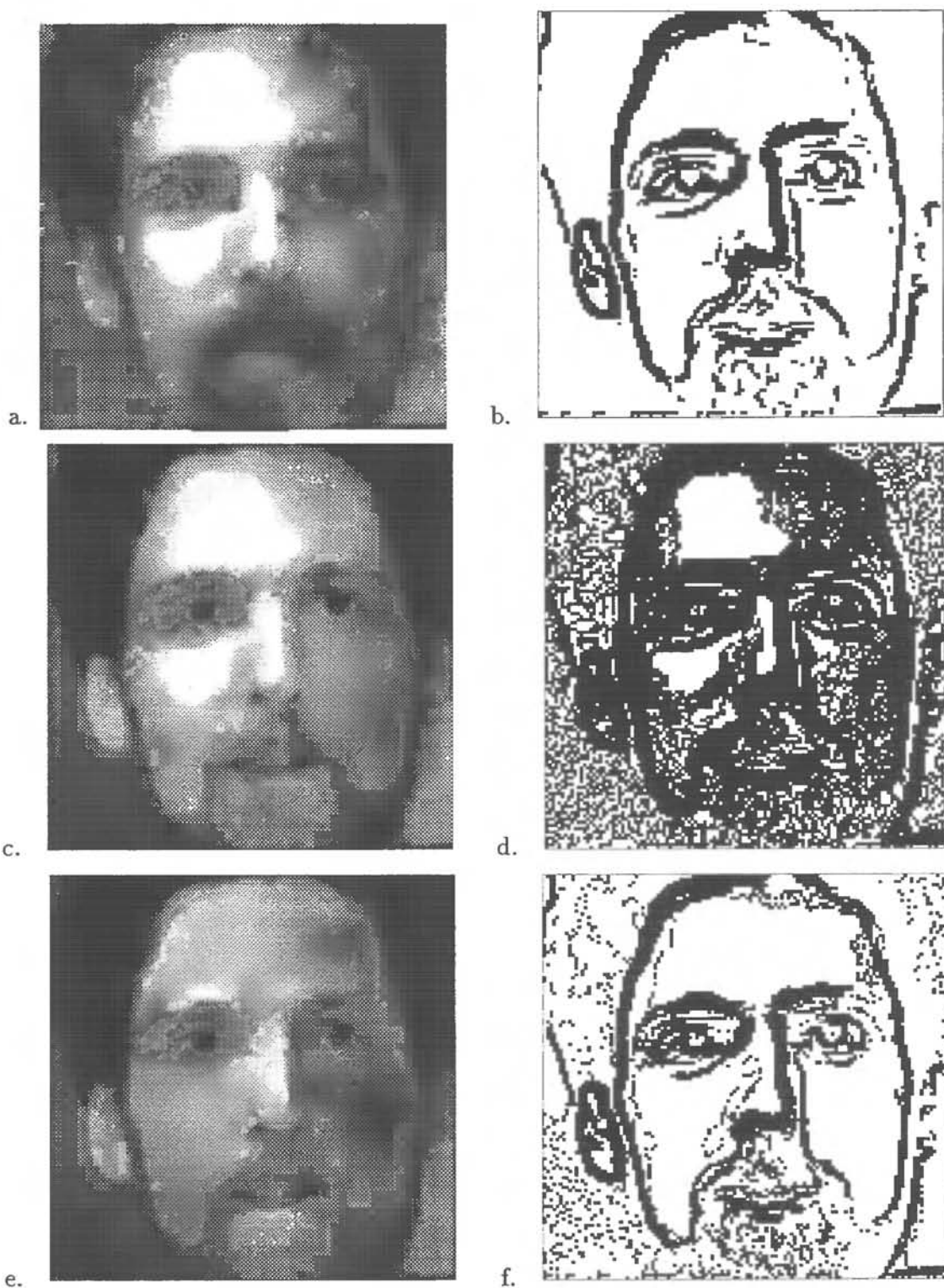

Figure 1: *(a) 8-bit image of 128 X 128 pixels. (b) The edge map for $\mu = 1.0$, $\gamma_{ij} = 100.0$. After 200 iterations and final $\beta = 25 \approx \infty$ (c) the approximated image for $\mu = 0.01$, $\gamma_{ij} = 1.0$ and $\eta = 0.0009$. (d) the corresponding sparse process (e) approximated image $\mu = 0.01$, $\gamma_{ij} = 1.0$ and $\eta = 0.0001$. (f) the corresponding sparse process.*